# On the Linear Convergence of the Proximal Gradient Method for Trace Norm Regularization

**Ke Hou, Zirui Zhou, Anthony Man–Cho So**
Department of Systems Engineering & Engineering Management
The Chinese University of Hong Kong
Shatin, N. T., Hong Kong
{khou,zrzhou,manchoso}@se.cuhk.edu.hk

**Zhi–Quan Luo**
Department of Electrical & Computer Engineering
University of Minnesota
Minneapolis, MN 55455, USA
luozq@ece.umn.edu

## Abstract

Motivated by various applications in machine learning, the problem of minimizing a convex smooth loss function with trace norm regularization has received much attention lately. Currently, a popular method for solving such problem is the proximal gradient method (PGM), which is known to have a sublinear rate of convergence. In this paper, we show that for a large class of loss functions, the convergence rate of the PGM is in fact linear. Our result is established without any strong convexity assumption on the loss function. A key ingredient in our proof is a new Lipschitzian error bound for the aforementioned trace norm–regularized problem, which may be of independent interest.

## 1 Introduction

The problem of finding a low–rank matrix that (approximately) satisfies a given set of conditions has recently generated a lot of interest in many communities. Indeed, such a problem arises in a wide variety of applications, including approximation algorithms [17], automatic control [5], matrix classification [20], matrix completion [6], multi–label classification [1], multi–task learning [2], network localization [7], subspace learning [24], and trace regression [9], just to name a few. Due to the combinatorial nature of the rank function, the task of recovering a matrix with the desired rank and properties is generally intractable. To circumvent this, a popular approach is to use the trace norm[1] (also known as the nuclear norm) as a surrogate for the rank function. Such an approach is quite natural, as the trace norm is the tightest convex lower bound of the rank function over the set of matrices with spectral norm at most one [13]. In the context of machine learning, the trace norm is typically used as a regularizer in the minimization of certain convex loss function. This gives rise to convex optimization problems of the form

$$\min_{X \in \mathbb{R}^{m \times n}} \left\{ F(X) = f(X) + \tau \|X\|_* \right\}, \tag{1}$$

where $f : \mathbb{R}^{m \times n} \to \mathbb{R}$ is the convex loss function, $\|X\|_*$ denotes the trace norm of $X$, and $\tau > 0$ is a regularization parameter. By standard results in convex optimization [4], the above formulation is tractable (i.e., polynomial–time solvable) for many choices of the loss function $f$. In practice,

however, one is often interested in settings where the decision variable $X$ is of high dimension. Thus, there has been much research effort in developing fast algorithms for solving (1) lately.

Currently, a popular method for solving (1) is the proximal gradient method (PGM), which exploits the composite nature of the objective function $F$ and certain smoothness properties of the loss function $f$ [8, 19, 11]. The attractiveness of PGM lies not only in its excellent numerical performance, but also in its strong theoretical convergence rate guarantees. Indeed, for the trace norm–regularized problem (1) with $f$ being convex and continuously differentiable and $\nabla f$ being Lipschitz continuous, the standard PGM will achieve an additive error of $O(1/k)$ in the optimal value after $k$ iterations. Moreover, this error can be reduced to $O(1/k^2)$ using acceleration techniques; see, e.g., [19]. The sublinear $O(1/k^2)$ convergence rate is known to be optimal if $f$ is simply given by a first–order oracle [12]. On the other hand, if $f$ is strongly convex, then the convergence rate can be improved to $O(c^k)$ for some $c \in (0, 1)$ (i.e., a linear convergence rate) [16]. However, in machine learning, the loss functions of interest are often highly structured and hence not just given by an oracle, but they are not necessarily strongly convex either. For instance, in matrix completion, a commonly used loss function is the square loss $f(\cdot) = \|\mathcal{A}(\cdot) - b\|_2^2/2$, where $\mathcal{A} : \mathbb{R}^{m \times n} \to \mathbb{R}^p$ is a linear measurement operator and $b \in \mathbb{R}^p$ is a given set of observations. Clearly, $f$ is not strongly convex when $\mathcal{A}$ has a non–trivial nullspace (or equivalently, when $\mathcal{A}$ is not injective). In view of this, it is natural to ask whether linear convergence of the PGM can be established for a larger class of loss functions.

In this paper, we take a first step towards answering this question. Specifically, we show that when the loss function $f$ takes the form $f(X) = h(\mathcal{A}(X))$, where $\mathcal{A} : \mathbb{R}^{m \times n} \to \mathbb{R}^p$ is an arbitrary linear operator and $h : \mathbb{R}^p \to \mathbb{R}$ is strictly convex with certain smoothness and curvature properties, the PGM for solving (1) has an asymptotic linear rate of convergence. Note that $f$ need not be strictly convex even if $h$ is, as $\mathcal{A}$ is arbitrary. Our result covers a wide range of loss functions used in the literature, such as square loss and logistic loss. Moreover, to the best of our knowledge, it is the first linear convergence result concerning the application of a first–order method to the trace norm–regularized problem (1) that does not require the strong convexity of $f$.

The key to our convergence analysis is a new Lipschitzian error bound for problem (1). Roughly, it says that the distance between a point $X \in \mathbb{R}^{m \times n}$ and the optimal solution set of (1) is on the order of the residual norm $\|\text{prox}_\tau(X - \nabla f(X)) - X\|_F$, where $\text{prox}_\tau$ is the proximity operator associated with the regularization term $\tau\|X\|_*$. Once we have such a bound, a routine application of the powerful analysis framework developed by Luo and Tseng [10] will yield the desired linear convergence result. Prior to this work, Lipschitzian error bounds for composite function minimization are available for cases where the non–smooth part either has a polyhedral epigraph (such as the $\ell_1$–norm) [23] or is the (sparse) group LASSO regularization [22, 25]. However, the question of whether a similar bound holds for trace norm regularization has remained open, despite its apparent similarity to $\ell_1$–norm regularization. Indeed, unlike the $\ell_1$–norm, the trace norm has a non–polyhedral epigraph; see, e.g., [18]. Moreover, the existing approach for establishing error bounds for $\ell_1$–norm or (sparse) group LASSO regularization is based on splitting the decision variables into groups, where variables from different groups do not interfere with one another, so that each group can be analyzed separately. However, the trace norm of a matrix is determined by its singular values, and each of them depends on every single entry of the matrix. Thus, we cannot use the same splitting approach to analyze the entries of the matrix. To overcome the above difficulties, we make the crucial observation that if $\bar{X}$ is an optimal solution to (1), then both $\bar{X}$ and $-\nabla f(\bar{X})$ have the same set of left and right singular vectors; see Proposition 4.2. As a result, we can use matrix perturbation theory to get hold of the spectral structure of the points that are close to the optimal solution set. This in turn allows us to establish a Lipschitzian error bound for the trace norm–regularized problem (1), thereby resolving the aforementioned open question in the affirmative.

## 2  Preliminaries

### 2.1  Basic Setup

We consider the trace norm–regularized optimization problem (1), in which the loss function $f : \mathbb{R}^{m \times n} \to \mathbb{R}$ takes the form

$$f(X) = h(\mathcal{A}(X)), \tag{2}$$

where $\mathcal{A} : \mathbb{R}^{m \times n} \to \mathbb{R}^p$ is a linear operator and $h : \mathbb{R}^p \to \mathbb{R}$ is a function satisfying the following assumptions:

**Assumption 2.1**

    *(a) The effective domain of h, denoted by dom(h), is open and non–empty.*

    *(b) The function h is continuously differentiable with Lipschitz–continuous gradient on dom(h) and is strongly convex on any convex compact subset of dom(h).*

Note that Assumption 2.1(b) implies the strict convexity of $h$ on dom$(h)$ and the Lipschitz continuity of $\nabla f$. Now, let $\mathcal{X}$ denote the set of optimal solutions to problem (1). We make the following assumption concerning $\mathcal{X}$:

**Assumption 2.2** *The optimal solution set $\mathcal{X}$ is non–empty.*

The above assumptions can be justified in various applications. For instance, in matrix completion, the square loss $f(\cdot) = \|\mathcal{A}(\cdot) - b\|_2^2/2$ induced by the linear measurement operator $\mathcal{A}$ and the set of observations $b \in \mathbb{R}^p$ is of the form (2), with $h(\cdot) = \|(\cdot) - b\|_2^2/2$. Moreover, it is clear that such an $h$ satisfies Assumptions 2.1 and 2.2. In multi–task learning, the loss function takes the form $f(\cdot) = \sum_{t=1}^T \ell(\mathcal{A}_t(\cdot), y_t)$, where $T$ is the number of learning tasks, $\mathcal{A}_t : \mathbb{R}^{m \times n} \to \mathbb{R}^p$ is the linear operator defined by the input data for the $t$–th task, $y_t \in \mathbb{R}^p$ is the output data for the $t$–th task, and $\ell : \mathbb{R}^p \times \mathbb{R}^p \to \mathbb{R}$ measures the learning error. Note that $f$ can be put into the form (2), where $\mathcal{A} : \mathbb{R}^{m \times n} \to \mathbb{R}^{Tp}$ is given by $\mathcal{A}(X) = (\mathcal{A}_1(X), \mathcal{A}_2(X), \ldots, \mathcal{A}_T(X))$, and $h : \mathbb{R}^{Tp} \to \mathbb{R}$ is given by $h(z) = \sum_{t=1}^T \ell(z_t, y_t)$ with $z_t \in \mathbb{R}^p$ for $t = 1, \ldots, T$ and $z = (z_1, \ldots, z_T)$. Moreover, in the case where $\ell$ is, say, the square loss (i.e., $\ell(z_t, y_t) = \|z_t - y_t\|_2^2/2$) or the logistic loss (i.e., $\ell(z_t, y_t) = \sum_{i=1}^p \log(1 + \exp(-z_{ti}y_{ti}))$), it can be verified that Assumptions 2.1 and 2.2 hold.

## 2.2 Some Facts about the Optimal Solution Set

Since $f(\cdot) = h(\mathcal{A}(\cdot))$ by (2) and $h(\cdot)$ is strictly convex on dom$(h)$ by Assumption 2.1(b), it is easy to verify that the map $X \mapsto \mathcal{A}(X)$ is invariant over the optimal solution set $\mathcal{X}$. In other words, there exists a $\bar{z} \in$ dom$(h)$ such that for any $X^* \in \mathcal{X}$, we have $\mathcal{A}(X^*) = \bar{z}$. Thus, we can express $\mathcal{X}$ as

$$\mathcal{X} = \left\{ X \in \mathbb{R}^{m \times n} : \tau\|X\|_* = v^* - h(\bar{z}), \ \mathcal{A}(X) = \bar{z} \right\},$$

where $v^* > -\infty$ is the optimal value of (1). In particular, $\mathcal{X}$ is a non–empty convex compact set. This implies that every $X \in \mathbb{R}^{m \times n}$ has a unique projection $\bar{X} \in \mathcal{X}$ onto $\mathcal{X}$, which is given by the solution to the following optimization problem:

$$\text{dist}(X, \mathcal{X}) = \min_{Y \in \mathcal{X}} \|X - Y\|_F.$$

In addition, since $\mathcal{X}$ is bounded and $F$ is convex, it follows from [14, Corollary 8.7.1] that the level set $\{X \in \mathbb{R}^{m \times n} : F(X) \leq \zeta\}$ is bounded for any $\zeta \in \mathbb{R}$.

## 2.3 Proximal Gradient Method and the Residual Map

To motivate the PGM for solving (1), we recall an alternative characterization of the optimal solution set $\mathcal{X}$. Consider the proximity operator prox$_\tau : \mathbb{R}^{m \times n} \to \mathbb{R}^{m \times n}$, which is defined as

$$\text{prox}_\tau(X) = \arg\min_{Z \in \mathbb{R}^{m \times n}} \left\{ \tau\|Z\|_* + \frac{1}{2}\|X - Z\|_F^2 \right\}. \tag{3}$$

By comparing the optimality conditions for (1) and (3), it is immediate that a solution $X^* \in \mathbb{R}^{m \times n}$ is optimal for (1) if and only if it satisfies the following fixed–point equation:

$$X^* = \text{prox}_\tau(X^* - \nabla f(X^*)). \tag{4}$$

This naturally lead to the following PGM for solving (1):

$$\begin{cases} Y^{k+1} &= X^k - \alpha_k \nabla f(X^k), \\ X^{k+1} &= \text{prox}_{\tau\alpha_k}(Y^{k+1}), \end{cases} \tag{5}$$

where $\alpha_k > 0$ is the step size in the $k$–th iteration, for $k = 0, 1, \ldots$; see, e.g., [8, 19, 11]. As is well–known, the proximity operator defined above can be expressed in terms of the so–called matrix

shrinkage operator. To describe this result, we introduce some definitions. Let $\mu > 0$ be given. The non–negative vector shrinkage operator $s_\mu : \mathbb{R}_+^p \to \mathbb{R}_+^p$ is defined as $(s_\mu(z))_i = \max\{0, z_i - \mu\}$, where $i = 1, \ldots, p$. The matrix shrinkage operator $S_\mu : \mathbb{R}^{m \times n} \to \mathbb{R}^{m \times n}$ is defined as $S_\mu(X) = U\Sigma_\mu V^T$, where $X = U\Sigma V^T$ is the singular value decomposition of $X$ with $\Sigma = \mathrm{Diag}(\sigma(X))$ and $\sigma(X)$ being the vector of singular values of $X$, and $\Sigma_\mu = \mathrm{Diag}(s_\mu(\sigma(X)))$. Then, it can be shown that

$$\mathrm{prox}_\tau(X) = S_\tau(X); \tag{6}$$

see, e.g., [11, Theorem 3].

Our goal in this paper is to study the convergence rate of the PGM (5). Towards that end, we need a measure to quantify its progress towards optimality. One natural candidate would be $\mathrm{dist}(\cdot, \mathcal{X})$, the distance to the optimal solution set $\mathcal{X}$. Despite its intuitive appeal, such a measure is hard to compute or analyze. In view of (4) and (6), a reasonable alternative would be the norm of the residual map $R : \mathbb{R}^{m \times n} \to \mathbb{R}^{m \times n}$, which is defined as

$$R(X) = S_\tau(X - \nabla f(X)) - X. \tag{7}$$

Intuitively, the residual map measures how much a solution $X \in \mathbb{R}^{m \times n}$ violates the optimality condition (4). In particular, $X$ is an optimal solution to (1) if and only if $R(X) = \mathbf{0}$. However, since $\|R(\cdot)\|_F$ is only a surrogate of $\mathrm{dist}(\cdot, \mathcal{X})$, we need to establish a relationship between them. This motivates the development of a so–called *error bound* for problem (1).

## 3 Main Results

Key to our convergence analysis of the PGM (5) is the following error bound for problem (1), which constitutes the main contribution of this paper:

**Theorem 3.1 (Error Bound for Trace Norm Regularization)** *Suppose that in problem (1), $f$ is of the form (2), and Assumptions 2.1 and 2.2 are satisfied. Then, for any $\zeta \geq v^*$, there exist constants $\eta > 0$ and $\epsilon > 0$ such that*

$$\mathrm{dist}(X, \mathcal{X}) \leq \eta \|R(X)\|_F \quad \text{whenever} \quad F(X) \leq \zeta, \ \|R(X)\|_F \leq \epsilon. \tag{8}$$

Armed with Theorem 3.1 and some standard properties of the PGM (5), we can apply the convergence analysis framework developed by Luo and Tseng [10] to establish the linear convergence of (5). Recall that a sequence of vectors $\{w^k\}_{k \geq 0}$ is said to converge *Q–linearly* (resp. *R–linearly*) to a vector $w^\infty$ if there exist an index $K \geq 0$ and a constant $\rho \in (0, 1)$ such that $\|w^{k+1} - w^\infty\|_2 / \|w^k - w^\infty\|_2 \leq \rho$ for all $k \geq K$ (resp. if there exist constants $\gamma > 0$ and $\rho \in (0, 1)$ such that $\|w^k - w^\infty\|_2 \leq \gamma \cdot \rho^k$ for all $k \geq 0$).

**Theorem 3.2 (Linear Convergence of the Proximal Gradient Method)** *Suppose that in problem (1), $f$ is of the form (2), and Assumptions 2.1 and 2.2 are satisfied. Moreover, suppose that the step size $\alpha_k$ in the PGM (5) satisfies $0 < \underline{\alpha} < \alpha_k < \bar{\alpha} < 1/L_f$ for $k = 0, 1, 2, \ldots$, where $L_f$ is the Lipschitz constant of $\nabla f$, and $\underline{\alpha}, \bar{\alpha}$ are given constants. Then, the sequence of solutions $\{X^k\}_{k \geq 0}$ generated by the PGM (5) converges R–linearly to an element in the optimal solution set $\mathcal{X}$, and the associated sequence of objective values $\{F(X^k)\}_{k \geq 0}$ converges Q–linearly to the optimal value $v^*$.*

Proof. Under the given setting, it can be shown that there exist scalars $\kappa_1, \kappa_2, \kappa_3 > 0$, which depend on $\underline{\alpha}, \bar{\alpha}$, and $L_f$, such that

$$F(X^k) - F(X^{k+1}) \geq \kappa_1 \|X^k - X^{k+1}\|_F^2, \tag{9}$$

$$F(X^{k+1}) - v^* \leq \kappa_2 \left[ (\mathrm{dist}(X^k, \mathcal{X}))^2 + \|X^{k+1} - X^k\|_F^2 \right], \tag{10}$$

$$\|R(X^k)\|_F \leq \kappa_3 \|X^k - X^{k+1}\|_F; \tag{11}$$

see the supplementary material. Since $\{F(X^k)\}_{k \geq 0}$ is a monotonically decreasing sequence by (9) and $F(X^k) \geq v^*$ for all $k \geq 0$, we conclude, again by (9), that $X^k - X^{k+1} \to \mathbf{0}$. This, together with (11), implies that $R(X^k) \to \mathbf{0}$. Thus, by (9), (10) and Theorem 3.1, there exist an index $K \geq 0$ and a constant $\kappa_4 > 0$ such that for all $k \geq K$,

$$F(X^{k+1}) - v^* \leq \kappa_4 \|X^k - X^{k+1}\|_F^2 \leq \frac{\kappa_4}{\kappa_1}(F(X^k) - F(X^{k+1})).$$

It follows that

$$F(X^{k+1}) - v^* \leq \frac{\kappa_4}{\kappa_1 + \kappa_4}(F(X^k) - v^*), \tag{12}$$

which establishes the Q–linear convergence of $\{F(X^k)\}_{k\geq 0}$ to $v^*$. Using (9) and (12), we can show that $\{\|X^{k+1} - X^k\|_F^2\}_{k\geq 0}$ converges R–linearly to 0, which, together with (11), implies that $\{X^k\}_{k\geq 0}$ converges R–linearly to a point in $\mathcal{X}$; see the supplementary material. $\qquad\square$

## 4 Proof of the Error Bound

The structure of our proof of Theorem 3.1 largely follows that laid out in [22, Section 6]. However, as explained in Section 1, some new ingredients are needed in order to analyze the spectral properties of a point that is close to the optimal solution set $\mathcal{X}$. Before we proceed, let us set up the notation that will be used in the proof. Let $L > 0$ denote the Lipschitz constant of $\nabla h$ and $\partial\|\cdot\|_*$ denote the subdifferential of $\|\cdot\|_*$. Given a sequence $\{X^k\}_{k\geq 0} \in \mathbb{R}^{m\times n}\backslash\mathcal{X}$, define

$$
\begin{aligned}
R^k = R(X^k), \quad \bar{X}^k = \arg\min_{Y\in\mathcal{X}}\|X^k - Y\|_F, \quad \delta_k = \|X^k - \bar{X}^k\|_F, \\
z^k = \mathcal{A}(X^k), \quad G^k = \nabla f(X^k) = \mathcal{A}^*(\nabla h(z^k)), \quad \bar{G} = \mathcal{A}^*(\nabla h(\bar{z})),
\end{aligned}
\tag{13}
$$

where $\mathcal{A}^*$ is the adjoint operator of $\mathcal{A}$. The crux of the proof of Theorem 3.1 is the following lemma:

**Lemma 4.1** *Under the setting of Theorem 3.1, suppose that there exists a convergent sequence* $\{X^k\}_{k\geq 0} \in \mathbb{R}^{m\times n}\backslash\mathcal{X}$ *satisfying*

$$F(X^k) \leq \zeta \text{ for all } k \geq 0, \quad R^k \to \mathbf{0}, \quad \frac{R^k}{\delta_k} \to \mathbf{0}. \tag{14}$$

*Then, the following hold:*

    *(a)* (Asymptotic Optimality) *The limit point $\bar{X}$ of $\{X^k\}_{k\geq 0}$ belongs to $\mathcal{X}$.*

    *(b)* (Bounded Iterates) *There exists a convex compact subset $\mathcal{Z}$ of dom$(h)$ such that $z^k, \bar{z} \in \mathcal{Z}$ for all $k \geq 0$. Consequently, there exists a constant $\sigma \in (0, L]$ such that for all $k \geq 0$,*

$$(\nabla h(z^k) - \nabla h(\bar{z}))^T(z^k - \bar{z}) \geq \sigma\|z^k - \bar{z}\|_2^2. \tag{15}$$

    *(c)* (Restricted Invertibility) *There exists a constant $\kappa > 0$ such that*

$$\|X^k - \bar{X}^k\|_F \leq \kappa\|z^k - \bar{z}\|_2 = \kappa\|\mathcal{A}(X^k - \bar{X}^k)\|_2 \quad \text{for all } k \geq 0. \tag{16}$$

It is clear that $\|\mathcal{A}(X^k - \bar{X}^k)\|_2 \leq \|\mathcal{A}\|\cdot\|X^k - \bar{X}^k\|_F$, where $\|\mathcal{A}\| = \sup_{\|Y\|_F=1}\|\mathcal{A}(Y)\|_2$ is the spectral norm of $\mathcal{A}$. Thus, the key element in Lemma 4.1 is the restricted invertibility property (16). For the sake of continuity, let us proceed to prove Theorem 3.1 by assuming the validity of Lemma 4.1.

Proof. [Theorem 3.1] We argue by contradiction. Suppose that there exists $\zeta \geq v^*$ such that (8) fails to hold for all $\eta > 0$ and $\epsilon > 0$. Then, there exists a sequence $\{X^k\}_{k\geq 0} \in \mathbb{R}^{m\times n}\backslash\mathcal{X}$ satisfying (14). Since $\{X \in \mathbb{R}^{m\times n} : F(X) \leq \zeta\}$ is bounded (see Section 2.2), by passing to a subsequence if necessary, we may assume that $\{X^k\}_{k\geq 0}$ converges to some $\bar{X}$. Hence, the premises of Lemma 4.1 are satisfied. Now, by Fermat's rule [15, Theorem 10.1], for each $k \geq 0$,

$$R^k \in \arg\min_D \left\{\langle G^k + R^k, D\rangle + \tau\|X^k + D\|_*\right\}. \tag{17}$$

Hence, we have

$$\langle G^k + R^k, R^k\rangle + \tau\|X^k + R^k\|_* \leq \langle G^k + R^k, \bar{X}^k - X^k\rangle + \tau\|\bar{X}^k\|_*.$$

Since $\bar{X}^k \in \mathcal{X}$ and $\nabla f(\bar{X}^k) = \bar{G}$, we also have $-\bar{G} \in \tau\partial\|\bar{X}^k\|_*$, which implies that

$$\tau\|\bar{X}^k\|_* \leq \langle \bar{G}, X^k + R^k - \bar{X}^k\rangle + \tau\|X^k + R^k\|_*.$$

Adding the two inequalities above and simplifying yield

$$\langle G^k - \bar{G}, X^k - \bar{X}^k\rangle + \|R^k\|_F^2 \leq \langle \bar{G} - G^k, R^k\rangle + \langle R^k, \bar{X}^k - X^k\rangle. \tag{18}$$

Since $z^k = \mathcal{A}(X^k)$ and $\bar{z} = \mathcal{A}(\bar{X}^k)$, by Lemma 4.1(b,c),

$$\langle G^k - \bar{G}, X^k - \bar{X}^k \rangle = (\nabla h(z^k) - \nabla h(\bar{z}))^T (z^k - \bar{z}) \geq \sigma \|z^k - \bar{z}\|_2^2 \geq \frac{\sigma}{\kappa^2} \|X^k - \bar{X}^k\|_F^2. \quad (19)$$

Hence, it follows from (15), (18), (19) and the Lipschitz continuity of $\nabla h$ that

$$\begin{aligned}
\frac{\sigma}{\kappa^2} \|X^k - \bar{X}^k\|_F^2 + \|R^k\|_F^2 &\leq (\nabla h(\bar{z}) - \nabla h(z^k))^T \mathcal{A}(R^k) + \langle R^k, \bar{X}^k - X^k \rangle \\
&\leq L \|\mathcal{A}\|^2 \|X^k - \bar{X}^k\|_F \|R^k\|_F + \|X^k - \bar{X}^k\|_F \|R^k\|_F.
\end{aligned}$$

In particular, this implies that

$$\frac{\sigma}{\kappa^2} \|X^k - \bar{X}^k\|_F^2 \leq (L\|\mathcal{A}\|^2 + 1)\|X^k - \bar{X}^k\|_F \|R^k\|_F$$

for all $k \geq 0$, which, upon dividing both sides by $\|X^k - \bar{X}^k\|_F$, yields a contradiction to (14). $\quad\square$

### 4.1 Proof of Lemma 4.1

We now return to the proof of Lemma 4.1. Since $R^k \to \mathbf{0}$ by (14) and $R$ is continuous, we have $R(\bar{X}) = \mathbf{0}$, which implies that $\bar{X} \in \mathcal{X}$. This establishes (a). To prove (b), observe that due to (a), the sequence $\{X^k\}_{k \geq 0}$ is bounded. Hence, the sequence $\{\mathcal{A}(X^k)\}_{k \geq 0}$ is also bounded, which implies that the points $z^k = \mathcal{A}(X^k)$ and $\bar{z} = \mathcal{A}(\bar{X})$ lie in a convex compact subset $\mathcal{Z}$ of $\text{dom}(h)$ for all $k \geq 0$. The inequality (15) then follows from Assumption 2.1(b). Note that we have $\sigma \leq L$, as $\nabla h$ is Lipschitz continuous with parameter $L$.

To prove (c), we argue by contradiction. Suppose that (16) is false. Then, by further passing to a subsequence if necessary, we may assume that

$$\|\mathcal{A}(X^k) - \bar{z}\|_2 / \|X^k - \bar{X}^k\|_F \to 0. \quad (20)$$

In the sequel, we will also assume without loss of generality that $m \leq n$. The following proposition establishes a property of the optimal solution set $\mathcal{X}$ that will play a crucial role in our proof.

**Proposition 4.2** *Consider a fixed $\bar{X} \in \mathcal{X}$. Let $\bar{X} - \bar{G} = \bar{U} \left[ Diag(\bar{\sigma}) \quad \mathbf{0} \right] \bar{V}^T$ be the singular value decomposition of $\bar{X} - \bar{G}$, where $\bar{U} \in \mathbb{R}^{m \times m}$, $\bar{V} \in \mathbb{R}^{n \times n}$ are orthogonal matrices and $\bar{\sigma}$ is the vector of singular values of $\bar{X} - \bar{G}$. Then, the matrices $\bar{X}$ and $-\bar{G}$ can be simultaneously singular–value–decomposed by $\bar{U}$ and $\bar{V}$. Moreover, the set $\mathcal{X}_c \subset \mathcal{X}$, which is defined as*

$$\mathcal{X}_c = \left\{ X \in \mathcal{X} : X = \bar{U} \left[ Diag(\sigma(X)) \quad \mathbf{0} \right] \bar{V}^T \right\},$$

*is a non–empty convex compact set.*

By Proposition 4.2, for every $k \geq 0$, the point $X^k$ has a unique projection $\tilde{X}^k \in \mathcal{X}_c$ onto $\mathcal{X}_c$. Let

$$\gamma_k = \|X^k - \tilde{X}^k\|_F = \min_{Y \in \mathcal{X}_c} \|X^k - Y\|_F. \quad (21)$$

Since $\mathcal{X}_c \subset \mathcal{X}$, we have $\gamma_k = \|X^k - \tilde{X}^k\|_F \geq \|X^k - \bar{X}^k\|_F = \delta_k$. It follows from (20) that $\|\mathcal{A}(X^k) - \bar{z}\|_2 / \|X^k - \tilde{X}^k\|_F \to 0$. This is equivalent to $\mathcal{A}(Q^k) \to \mathbf{0}$, where

$$Q^k = \frac{X^k - \tilde{X}^k}{\gamma_k} \quad \text{for all } k \geq 0. \quad (22)$$

In particular, we have $\|Q^k\|_F = 1$ for all $k \geq 0$. By further passing to a subsequence if necessary, we will assume that $\{Q^k\}_{k \geq 0}$ converges to some $\bar{Q}$. Clearly, we have $\mathcal{A}(\bar{Q}) = \mathbf{0}$ and $\|\bar{Q}\|_F = 1$.

#### 4.1.1 Decomposing $\bar{Q}$

Our goal now is to show that for $k$ sufficiently large and $\epsilon > 0$ sufficiently small, the point $\hat{X} = \tilde{X}^k + \epsilon \bar{Q}$ belongs to $\mathcal{X}_c$ and is closer to $X^k$ than $\tilde{X}^k$ is to $X^k$. This would then contradict the fact that $\tilde{X}^k$ is the projection of $X^k$ onto $\mathcal{X}_c$. To begin, let $\sigma^k$ be the vector of singular values of $X^k - G^k$. Since $X^k - G^k \to \bar{X} - \bar{G}$, the sequence $\{\sigma^k\}_{k \geq 0}$ is bounded. Hence, for $i = 1, \ldots, m$, by passing to a subsequence if necessary, we can classify the sequence $\{\sigma_i^k\}_{k \geq 0}$ into one of the following three cases: (A) $\sigma_i^k \leq \tau$ for all $k \geq 0$; (B) $\sigma_i^k > \tau$ and $\sigma_i(\tilde{X}^k) > 0$ for all $k \geq 0$; (C) $\sigma_i^k > \tau$ and $\sigma_i(\tilde{X}^k) = 0$ for all $k \geq 0$. The following proposition gives the key structural properties of $\bar{Q}$ that will lead to the desired contradiction:

**Proposition 4.3** *The matrix $\bar{Q}$ admits the decomposition $\bar{Q} = \bar{U}\,[Diag(\lambda) \quad \mathbf{0}]\,\bar{V}^T$, where*

$$\lambda_i \begin{cases} = -\lim\limits_{k\to\infty} \dfrac{\sigma_i(\tilde{X}^k)}{\gamma_k} \leq 0 & \text{in Case (A)}, \\[2mm] \in \mathbb{R} & \text{in Case (B)}, \\[2mm] > 0 & \text{in Case (C)}, \end{cases} \qquad \text{for } i = 1, \ldots, m.$$

It should be noted that the decomposition given in Proposition 4.3 is not necessarily the singular value decomposition of $\bar{Q}$, as $\lambda$ could have negative components. A proof of Proposition 4.3 can be found in the supplementary material.

### 4.1.2 Completing the Proof

Armed with Proposition 4.3, we are now ready to complete the proof of Lemma 4.1(c). Since $Q^k \neq \mathbf{0}$ for all $k \geq 0$, it follows from (22) that $\langle X^k - \tilde{X}^k, \bar{Q}\rangle > 0$ for all $k$ sufficiently large. Fix any such $k$ and let $\hat{X} = \tilde{X}^k + \epsilon\bar{Q}$, where $\epsilon > 0$ is a parameter to be determined. Since $\mathcal{A}(\bar{Q}) = \mathbf{0}$, it follows from (13) that $\nabla f(\hat{X}) = \nabla f(\tilde{X}^k) = \bar{G}$. Moreover, since $\tilde{X}^k \in \mathcal{X}_c$, by the optimality condition (4) and Proposition 4.2, we have

$$\max\left\{0, \sigma_i(\tilde{X}^k) + \sigma_i(-\bar{G}) - \tau\right\} = \sigma_i(\tilde{X}^k) \quad \text{for } i = 1, \ldots, m. \tag{23}$$

Now, we claim that for $\epsilon > 0$ sufficiently small, $\hat{X}$ satisfies

$$S_\tau(\hat{X} - \bar{G})\bar{v}_i = \hat{X}\bar{v}_i \qquad\qquad \text{for } i = 1, \ldots, n, \tag{24}$$
$$\bar{u}_i^T S_\tau(\hat{X} - \bar{G}) = \bar{u}_i^T \hat{X} \qquad\qquad \text{for } i = 1, \ldots, m,$$

where $\bar{u}_i$ (resp. $\bar{v}_i$) is the $i$–th column of $\bar{U}$ (resp. $\bar{V}$). This would then imply that $\hat{X} \in \mathcal{X}_c$. To prove the claim, observe that for $i = m + 1, \ldots, n$, both sides of (24) are equal to $\mathbf{0}$. Moreover, since $\tilde{X}^k \in \mathcal{X}_c$, Propositions 4.2 and 4.3 give

$$\hat{X} - \bar{G} = \bar{U}\left[\mathrm{Diag}(\sigma(\tilde{X}^k) + \epsilon\lambda + \sigma(-\bar{G})) \quad \mathbf{0}\right]\bar{V}^T.$$

Thus, it suffices to show that for $\epsilon > 0$ sufficiently small,

$$\sigma_i(\tilde{X}^k) + \epsilon\lambda_i + \sigma_i(-\bar{G}) \geq 0 \qquad\qquad \text{for } i = 1, \ldots, m, \tag{25}$$
$$s_\tau(\sigma_i(\tilde{X}^k) + \epsilon\lambda_i + \sigma_i(-\bar{G})) = \sigma_i(\tilde{X}^k) + \epsilon\lambda_i \qquad\qquad \text{for } i = 1, \ldots, m. \tag{26}$$

Towards that end, fix an index $i = 1, \ldots, m$ and consider the three cases defined in Section 4.1.1:

**Case (A).** If $\sigma_i(\tilde{X}^k) = 0$ for all $k$ sufficiently large, then Proposition 4.3 gives $\lambda_i = 0$. Moreover, we have $\sigma_i(-\bar{G}) \leq \tau$ by (23). This implies that both (25) and (26) are satisfied for any choice of $\epsilon > 0$. On the other hand, if $\sigma_i(\tilde{X}^k) > 0$ for all $k$ sufficiently large, then Proposition 4.3 gives $\lambda_i < 0$. Moreover, we have $\sigma_i(-\bar{G}) = \tau$ by (23). By choosing $\epsilon > 0$ so that $\sigma_i(\tilde{X}^k) + \epsilon\lambda_i \geq 0$, we can guarantee that both (25) and (26) are satisfied.

**Case (B).** Since $\sigma_i(\tilde{X}^k) > 0$ for all $k \geq 0$, we have $\sigma_i(-\bar{G}) = \tau$ by (23). Hence, both (25) and (26) can be satisfied by choosing $\epsilon > 0$ so that $\sigma_i(\tilde{X}^k) + \epsilon\lambda_i \geq 0$.

**Case (C).** By Proposition 4.2, we have $\bar{X} \in \mathcal{X}_c$. Since $X^k \to \bar{X}$ and $\gamma_k = \|X^k - \tilde{X}^k\|_F \leq \|X^k - \bar{X}\|_F$, we have $\tilde{X}^k \to \bar{X}$ as well. It follows that $\sigma_i(\bar{X}) = 0$, as $\sigma_i(\tilde{X}^k) = 0$ for all $k \geq 0$ by assumption. Now, since $X^k - G^k \to \bar{X} - \bar{G}$ and $\sigma_i^k > \tau$, we have $\bar{\sigma}_i \geq \tau$. Thus, Proposition 4.2 implies that $\tau \leq \bar{\sigma}_i = \sigma_i(\bar{X} - \bar{G}) = \sigma_i(\bar{X}) + \sigma_i(-\bar{G}) = \sigma_i(-\bar{G})$. This, together with (23), yields $\sigma_i(-\bar{G}) = \tau$. Since $\lambda_i > 0$ by Proposition 4.3, we conclude that both (25) and (26) can be satisfied by any choice of $\epsilon > 0$.

Thus, in all three cases, the claim is established. In particular, we have $\hat{X} \in \mathcal{X}_c$. This, together with $\langle X^k - \tilde{X}^k, \bar{Q}\rangle > 0$ and $\|\bar{Q}\|_F = 1$, yields

$$\|X^k - \hat{X}\|_F^2 = \|X^k - \tilde{X}^k - \epsilon\bar{Q}\|_F^2 = \|X^k - \tilde{X}^k\|_F^2 - 2\epsilon\langle X^k - \tilde{X}^k, \bar{Q}\rangle + \epsilon^2 < \|X^k - \tilde{X}^k\|_F^2$$

for $\epsilon > 0$ sufficiently small, which contradicts the fact that $\tilde{X}^k$ is the projection of $X^k$ onto $\mathcal{X}_c$. This completes the proof of Lemma 4.1(c).

# 5 Numerical Experiments

In this section, we complement our theoretical results by testing the numerical performance of the PGM (5) on two problems: matrix completion and matrix classification.

**Matrix Completion:** We randomly generate an $n \times n$ matrix $M$ with a prescribed rank $r$. Then, we fix a sampling ratio $\theta \in (0, 1]$ and sample $p = \lfloor \theta n^2 \rfloor$ entries of $M$ uniformly at random. This induces a sampling operator $\mathcal{P} : \mathbb{R}^{m \times n} \to \mathbb{R}^p$ and an observation vector $b \in \mathbb{R}^p$. In our experiments, we fix the rank $r = 3$ and use the square loss $f(\cdot) = \|\mathcal{P}(\cdot) - b\|_2^2 / 2$ with regularization parameter $\mu = 1$ in problem (1). We then solve the resulting problem for different values of $n$ and $\theta$ using the PGM (5) with a fixed step size $\alpha = 1$. We stop the algorithm when $F(X^k) - F(X^{k+1}) < 10^{-8}$. Figure 1 shows the semi–log plots of the error in objective value and the error in solution against the number of iterations. It can be seen that as long as the iterates are close enough to the optimal set, both the objective values and the solutions converge linearly.

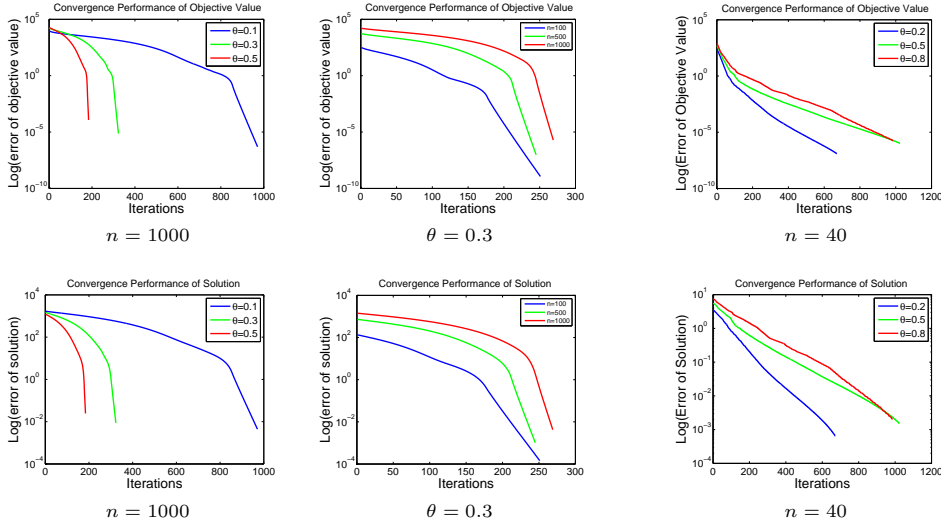 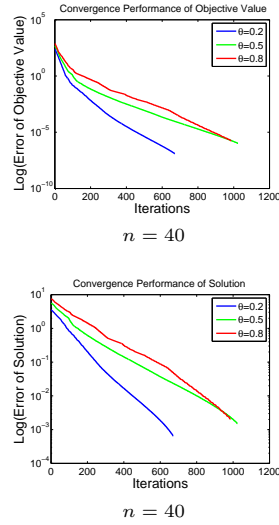

Figure 1: Matrix Completion                 Figure 2: Matrix Classification

**Matrix Classification:** We consider a matrix classification problem under the setting described in [21]. Specifically, we first randomly generate a low-rank matrix classifier $X^*$, which is an $n \times n$ symmetric matrix of rank $r$. Then, we specify a sampling ratio $\theta \in (0, 1]$ and sample $p = \lfloor \theta n^2 \rfloor / 2$ independent $n \times n$ symmetric matrices $W_1, \ldots, W_p$ from the standard Wishart distribution with $n$ degrees of freedom. The label of $W_i$, denoted by $y_i$, is given by $\mathrm{sgn}(\langle X^*, W_i \rangle)$. In our experiments, we fix the rank $r = 3$, the dimension $n = 40$, and use the logistic loss $f(\cdot) = \sum_{i=1}^p \log(1 + \exp(-y_i \langle \cdot, W_i \rangle))$ with regularization parameter $\mu = 1$ in problem (1). Since a good lower bound on the Lipschitz constant $L_f$ of $\nabla f$ is not readily available in this case, a backtracking line search was adopted at each iteration to achieve an acceptable step size; see, e.g., [3]. We stop the algorithm when $F(X^k) - F(X^{k+1}) < 10^{-6}$. Figure 2 shows the convergence performance of the PGM (5) as $\theta$ varies. Again, it can be seen that both the objective values and the solutions converge linearly.

# 6 Conclusion

In this paper, we have established the linear convergence of the PGM for solving a class of trace norm–regularized problems. Our convergence result does not require the objective function to be strongly convex and is applicable to many settings in machine learning. The key technical tool in the proof is a Lipschitzian error bound for trace norm–regularized problems, which could be of independent interest. A future direction is to study error bounds for more general matrix norm–regularized problems and their implications on the convergence rates of first–order methods.

**Acknowledgments** The authors would like to thank the anonymous reviewers for their careful reading of the manuscript and insightful comments. The research of A. M.–C. So is supported in part by a gift grant from Microsoft Research Asia.

## Footnotes

[1]Recall that the trace norm of a matrix is defined as the sum of its singular values.

# References

[1] Y. Amit, M. Fink, N. Srebro, and S. Ullman. Uncovering Shared Structures in Multiclass Classification. In *Proc. 24th ICML*, pages 17–24, 2007.

[2] A. Argyriou, T. Evgeniou, and M. Pontil. Convex Multi–Task Feature Learning. *Mach. Learn.*, 73(3):243–272, 2008.

[3] A. Beck and M. Teboulle. A Fast Iterative Shrinkage–Thresholding Algorithm for Linear Inverse Problems. *SIAM J. Imaging Sci.*, 2(1):183–202, 2009.

[4] A. Ben-Tal and A. Nemirovski. *Lectures on Modern Convex Optimization: Analysis, Algorithms, and Engineering Applications*. MPS–SIAM Series on Optimization. Society for Industrial and Applied Mathematics, Philadelphia, Pennsylvania, 2001.

[5] M. Fazel, H. Hindi, and S. P. Boyd. A Rank Minimization Heuristic with Application to Minimum Order System Approximation. In *Proc. 2001 ACC*, pages 4734–4739, 2001.

[6] D. Gross. Recovering Low–Rank Matrices from Few Coefficients in Any Basis. *IEEE Trans. Inf. Theory*, 57(3):1548–1566, 2011.

[7] S. Ji, K.-F. Sze, Z. Zhou, A. M.-C. So, and Y. Ye. Beyond Convex Relaxation: A Polynomial–Time Non–Convex Optimization Approach to Network Localization. In *Proc. 32nd IEEE INFOCOM*, pages 2499–2507, 2013.

[8] S. Ji and J. Ye. An Accelerated Gradient Method for Trace Norm Minimization. In *Proc. 26th ICML*, pages 457–464, 2009.

[9] V. Koltchinskii, K. Lounici, and A. B. Tsybakov. Nuclear–Norm Penalization and Optimal Rates for Noisy Low–Rank Matrix Completion. *Ann. Stat.*, 39(5):2302–2329, 2011.

[10] Z.-Q. Luo and P. Tseng. Error Bounds and Convergence Analysis of Feasible Descent Methods: A General Approach. *Ann. Oper. Res.*, 46(1):157–178, 1993.

[11] S. Ma, D. Goldfarb, and L. Chen. Fixed Point and Bregman Iterative Methods for Matrix Rank Minimization. *Math. Program.*, 128(1–2):321–353, 2011.

[12] Yu. Nesterov. *Introductory Lectures on Convex Optimization: A Basic Course*. Kluwer Academic Publishers, Boston, 2004.

[13] B. Recht, M. Fazel, and P. A. Parrilo. Guaranteed Minimum–Rank Solutions of Linear Matrix Equations via Nuclear Norm Minimization. *SIAM Rev.*, 52(3):471–501, 2010.

[14] R. T. Rockafellar. *Convex Analysis*. Princeton Landmarks in Mathematics and Physics. Princeton University Press, Princeton, New Jersey, 1997.

[15] R. T. Rockafellar and R. J.-B. Wets. *Variational Analysis*, volume 317 of *Grundlehren der mathematischen Wissenschaften*. Springer–Verlag, Berlin Heidelberg, second edition, 2004.

[16] M. Schmidt, N. Le Roux, and F. Bach. Convergence Rates of Inexact Proximal–Gradient Methods for Convex Optimization. In *Proc. NIPS 2011*, pages 1458–1466, 2011.

[17] A. M.-C. So, Y. Ye, and J. Zhang. A Unified Theorem on SDP Rank Reduction. *Math. Oper. Res.*, 33(4):910–920, 2008.

[18] W. So. Facial Structures of Schatten $p$–Norms. *Linear and Multilinear Algebra*, 27(3):207–212, 1990.

[19] K.-C. Toh and S. Yun. An Accelerated Proximal Gradient Algorithm for Nuclear Norm Regularized Linear Least Squares Problems. *Pac. J. Optim.*, 6(3):615–640, 2010.

[20] R. Tomioka and K. Aihara. Classifying Matrices with a Spectral Regularization. In *Proc. of the 24th ICML*, pages 895–902, 2007.

[21] R. Tomioka, T. Suzuki, M. Sugiyama, and H. Kashima. A Fast Augmented Lagrangian Algorithm for Learning Low–Rank Matrices. In *Proc. 27th ICML*, pages 1087–1094, 2010.

[22] P. Tseng. Approximation Accuracy, Gradient Methods, and Error Bound for Structured Convex Optimization. *Math. Program.*, 125(2):263–295, 2010.

[23] P. Tseng and S. Yun. A Coordinate Gradient Descent Method for Nonsmooth Separable Minimization. *Math. Program.*, 117(1–2):387–423, 2009.

[24] M. White, Y. Yu, X. Zhang, and D. Schuurmans. Convex Multi–View Subspace Learning. In *Proc. NIPS 2012*, pages 1682–1690, 2012.

[25] H. Zhang, J. Jiang, and Z.-Q. Luo. On the Linear Convergence of a Proximal Gradient Method for a Class of Nonsmooth Convex Minimization Problems. *J. Oper. Res. Soc. China*, 1(2):163–186, 2013.

